# Neural Implementation of Motivated Behavior: Feeding in an Artificial Insect

**Randall D. Beer**[1,2] **and Hillel J. Chiel**[2]
Departments of [1]Computer Engineering and Science, and [2]Biology
and the Center for Automation and Intelligent Systems Research
Case Western Reserve University
Cleveland, OH 44106

## ABSTRACT

Most complex behaviors appear to be governed by internal motivational states or drives that modify an animal's responses to its environment. It is therefore of considerable interest to understand the neural basis of these motivational states. Drawing upon work on the neural basis of feeding in the marine mollusc *Aplysia*, we have developed a heterogeneous artificial neural network for controlling the feeding behavior of a simulated insect. We demonstrate that feeding in this artificial insect shares many characteristics with the motivated behavior of natural animals.

## 1   INTRODUCTION

While an animal's external environment certainly plays an extremely important role in shaping its actions, the behavior of even simpler animals is by no means solely reactive. The response of an animal to food, for example, cannot be explained only in terms of the physical stimuli involved. On two different occasions, the very same animal may behave in completely different ways when presented with seemingly identical pieces of food (e.g. hungrily consuming it in one case and ignoring or even avoiding it in another). To account for these differences, behavioral scientists hypothesize internal motivational states or drives which modulate an animal's response to its environment. These internal factors play a particularly important role in complex behavior, but are present to some degree in nearly all animal behavior. Behaviors which exhibit an extensive dependence on motivational variables are termed *motivated behaviors*.

While a rigorous definition is difficult to state, behaviors spoken of as motivated generally exhibit some subset of the following six characteristics (Kupfermann, 1974): (1) grouping and sequencing of component behaviors in time, (2) goal-directedness: the sequence of component behaviors generated can often be understood only by reference to some internal goal, (3) spontaneity: the behavior can occur in the absence of any recognizable eliciting stimuli, (4) changes in responsiveness: the effect of a motivational state varies depending upon an animal's level of arousal, (5) persistence: the behavior can greatly outlast any initiating stimulus, and (6) associative learning.

Motivational states are pervasive in mammalian behavior. However, they have also proven to be essential for explaining the behavior of simpler animals as well. Unfortunately, the explanatory utility of these internal factors is limited by the fact that they are hypothetical constructs, inferred by the theorist to intervene between stimulus and action in order to account for otherwise inexplicable responses. What might be the neural basis of these motivational states?

In order to explore this question, we have drawn upon work on the neural basis of feeding in the marine mollusc *Aplysia* to implement feeding in a simulated insect. Feeding is a prototypical motivated behavior in which attainment of the goal object (food) is clearly crucial to an animal's survival. In this case, the relevant motivational state is hunger. When an animal is hungry, it will exhibit a sequence of *appetitive behaviors* which serve to identify and properly orient the animal to food. Once food is available, *consummatory behaviors* are generated to ingest it. On the other hand, a satiated animal may ignore or even avoid sensory stimuli which suggest the presence of food (Kupfermann, 1974).

This effort is part of a larger project aimed at designing artificial nervous systems for the flexible control of complete autonomous agents (Beer, 1989). In addition to feeding, this artificial insect is currently capable of locomotion (Beer, Chiel, and Sterling, 1989; Chiel and Beer, 1989), wandering, and edge-following, and possesses a simple behavioral hierarchy as well. A central theme of this work has been the utilization of biologically-inspired architectures in our neural network designs. To support this capability, we make use of model neurons which capture some of the intrinsic properties of nerve cells.

The simulated insect and the environment in which it exists is designed as follows. The insect has six legs, and is capable of statically stable locomotion and turning. Its head contains a mouth which can open and close, and its mouth and two antennae possess tactile and chemical sensors. The insect possesses an internal energy supply which is depleted at a fixed rate. The simulated environment also contains unmovable obstacles and circular food patches. The food patches emit an odor whose intensity is proportional to the size of the patch. As this odor diffuses through the environment, its intensity falls off as the inverse square of the distance from the center of the patch. Whenever the insect's mouth closes over a patch of food, a fixed amount of energy is transferred from the patch to the insect.

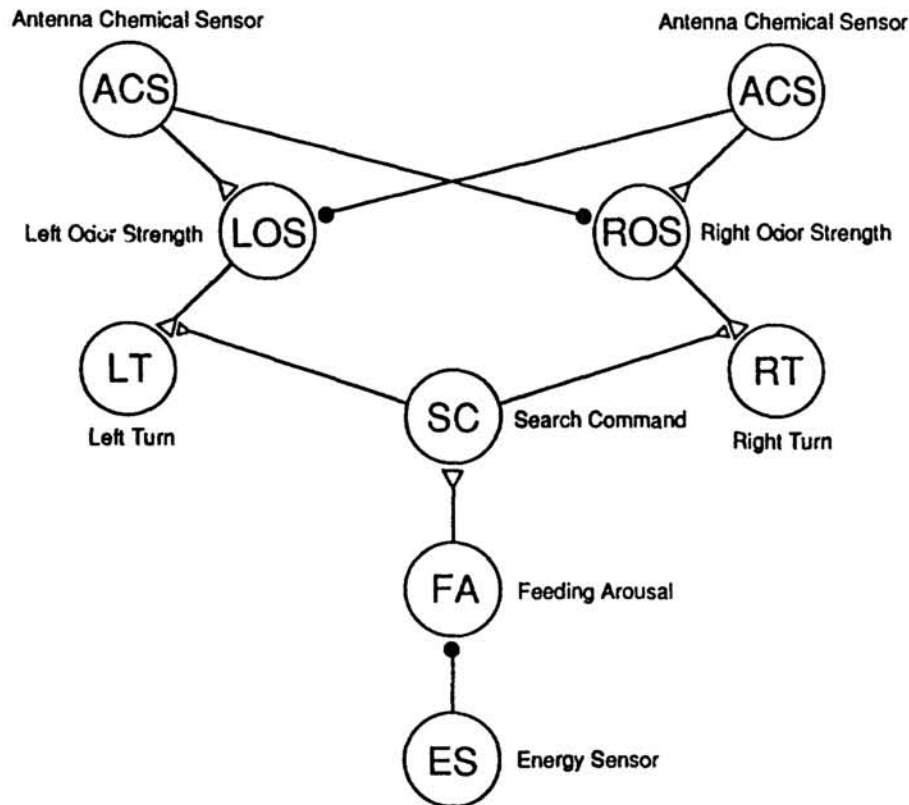

Figure 1: Appetitive Controller

## 2    APPETITIVE COMPONENT

The appetitive component of feeding is responsible for getting a hungry insect to a food patch. To accomplish this task, it utilizes the locomotion, wandering, and edge-following capabilities of the insect. The interactions between the neural circuitry underlying these behaviors and the feeding controller presented in this paper are described elsewhere (Beer, 1989). Assuming that the insect is already close enough to a food patch that the chemical sensors in its antennae can detect an odor signal, there are two separate issues which must be addressed by this phase of the behavior. First, the insect must use the information from the chemical sensors in its antennae to turn itself toward the food patch as it walks. Second, this orientation should only occur when the insect is actually in need of energy. Correspondingly, the appetitive neural controller (Figure 1) consists of two distinct components.

The orientation component is comprised of the upper six neurons in Figure 1. The odor signals detected by the chemical sensors in each antenna (ACS) are compared (by LOS and ROS), and the difference between them is used to generate a turn toward the stronger side by exciting the corresponding turn interneuron (LT or RT) by an amount proportional to the size of the difference. These turn interneurons connect to the motor neurons controlling the lateral extension of each front leg.

The second component is responsible for controlling whether or not the insect ac-

tually orients to a nearby patch of food. This decision depends upon its internal energy level, and is controlled by the bottom three neurons in Figure 1. Though the odor gradient is continuously being sensed, the connections to the turn interneurons are normally disabled, preventing access of this information to the motor apparatus which turns the insect. As the insect's energy level falls, however, so does the activity of its energy sensor (ES). This decreasing activity gradually releases the spontaneously active feeding arousal neuron (FA) from inhibition. When activity in FA becomes sufficient to fire the search command neuron (SC), the connections between the odor strength neurons and the turn neurons are enabled by gating connections from SC, and the insect begins to orient to food.

# 3    CONSUMMATORY COMPONENT

Once the appetitive controller has successfully oriented the insect to food, the consummatory component of the behavior is triggered. This phase consists of rhythmic biting movements which persist until sufficient food has been ingested. Like the appetitive phase, consummatory behavior should only be released when the insect is in need of energy. In addition, an animal's interest in feeding (its *feeding arousal*), may be a function of more than just its energy requirements. Other factors, such as the exposure of an animal to the taste, odor, or tactile sensations of food, can significantly increase its feeding arousal. This relationship between feeding and arousal, in which the very act of feeding further enhances an animal's interest in feeding, leads to a form of behavioral hysteresis. Once food is encountered, an animal may feed well beyond the internal energy requirements which initiated the behavior. In many animals, this hysteresis is thought to play a role in the patterning of feeding behavior into discrete meals rather than continuous grazing (Susswein, Weiss, and Kupfermann, 1978). At some point, of course, the ingested food must be capable of overriding the arousing effects of consummatory behavior, or the animal would never cease to feed.

The neural controller for the consummatory phase of feeding is shown in Figure 2. When chemical (MCS) and tactile (MTS) sensors in the mouth signal that food is present (FP), and the insect is sufficiently aroused to feeding (FA), the consummatory command neuron (CC) fires. The conjunction of tactile and chemical signals is required in order to prevent attempts to ingest nonfood patches and, due to the diffusion of odors, to prevent biting from beginning before the food patch is actually reached. Once CC fires, it triggers the bite pacemaker neuron (BP) to generate rhythmic bursts which cause a motor neuron (MO) to open and close the mouth. Because the threshold of the consummatory command neuron (CC) is somewhat lower than that of the search command neuron (SC), an insect which is not sufficiently aroused to orient to food may nevertheless consume food that is directly presented to its mouth.

The motor neuron controlling the mouth also makes an excitatory connection onto the feeding arousal neuron (FA), which in turn makes an excitatory modulatory synapse onto the connection between the consummatory command neuron (CC)

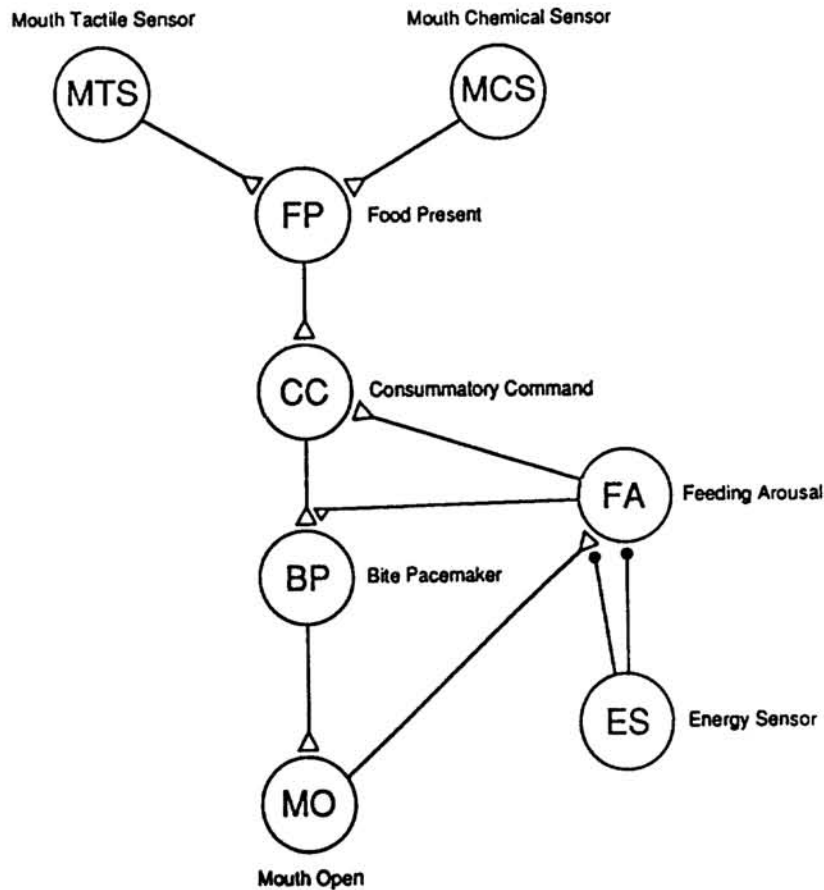

**Figure 2:** Consummatory Controller

and the bite pacemaker (BP). The net effect of these excitatory connections is a positive feedback loop: biting movements excite FA, which causes BP to cause more frequent biting movements, which further excites FA until its activity saturates. This neural positive feedback loop is inspired by work on the neural basis of feeding arousal maintenance in *Aplysia* (Weiss, Chiel, Koch, and Kupfermann, 1986).

As the insect consumes food, its energy level begins to rise. This leads to increased activity in ES which both directly inhibits FA, and also decreases the gain of the positive feedback loop via an inhibitory modulatory synapse onto the connection between MO and FA. At some point, these inhibitory effects will overcome the positive feedback and activity in FA will drop low enough to terminate the feeding behavior. This neural mechanism is based upon a similar one hypothesized to underlie satiation in *Aplysia* (Weiss, Chiel, and Kupfermann, 1986).

## 4    RESULTS

With the neural controllers described above, we have found that feeding behavior in the artificial insect exhibits four of the six characteristics of motivated behavior which were described by Kupfermann (1974):

**Grouping and sequencing of behavior in time.** When the artificial insect is "hungry", it generates appetitive and consummatory behaviors with the proper sequence, timing, and intensity in order to obtain food.

**Goal-Directedness.** Regardless of its environmental situation, a hungry insect will generate movements which serve to obtain food. Therefore, the behavior of a hungry insect can only be understood by reference to an internal goal. Due to the internal effects of the energy sensor (ES) and feeding arousal (FA) neurons on the controllers, the insect's external stimuli are insufficient to account for its behavior.

**Changes in responsiveness due to a change in internal state.** While a hungry insect will attempt to orient to and consume any nearby food, a satiated one will ignore it. In addition, once a hungry insect has consumed sufficient food, it will simply walk over the food patch which initially attracted it. We will examine the arousal and satiation of feeding in this artificial insect in more detail below.

**Persistence.** If a hungry insect is removed from food before it has fed to satiation, its feeding arousal will persist, and it will continue to exhibit feeding movements.

One technique that has been applied to the study of feeding arousal in natural animals is the examination of the time interval between successive bites as an animal feeds under various conditions. In *Aplysia*, for example, the interbite interval progressively decreases as an animal begins to feed (showing a build-up of arousal), and increases as the animal satiates. In addition, the rate of rise and fall of arousal depends upon the initial degree of satiation (Susswein, Weiss, and Kupfermann, 1978).

In order to examine the role of feeding arousal in the artificial insect, we performed a similar set of experiments. Food was directly presented to insects with differing degrees of initial satiation, and the time interval between successive bites was recorded for the entire resulting consummatory response. Above an energy level of approximately 80% of capacity, insects could not be induced to bite. Below this level, however, insects began to consume the food. As these insects fed, the interbite interval decreased as their feeding arousal built up until some minimum interval was achieved (Figure 3). The rate of build-up of arousal was slowest for those insects with the highest initial degree of satiation. In fact, an insect whose energy level was already 75% of capacity never achieved full arousal. As the feeding insects neared satiation, their interbite interval increased as arousal waned. It is interesting to note that, regardless of the initial degree of satiation, all insects in which biting was triggered fed until their energy stores were approximately 99% full. The appropriate number of bites to achieve this were generated in all cases.

What is the neural basis of these arousal and satiation phenomena? Clearly, the answer lies in the interactions between the internal energy sensor and the positive feedback loop mediated by the feeding arousal neuron, but the precise nature of the interaction is not at all clear from the qualitative descriptions of the neural controllers given earlier. In order to more carefully examine this interaction, we produced a phase plot of the activity in these two neurons under the experimental

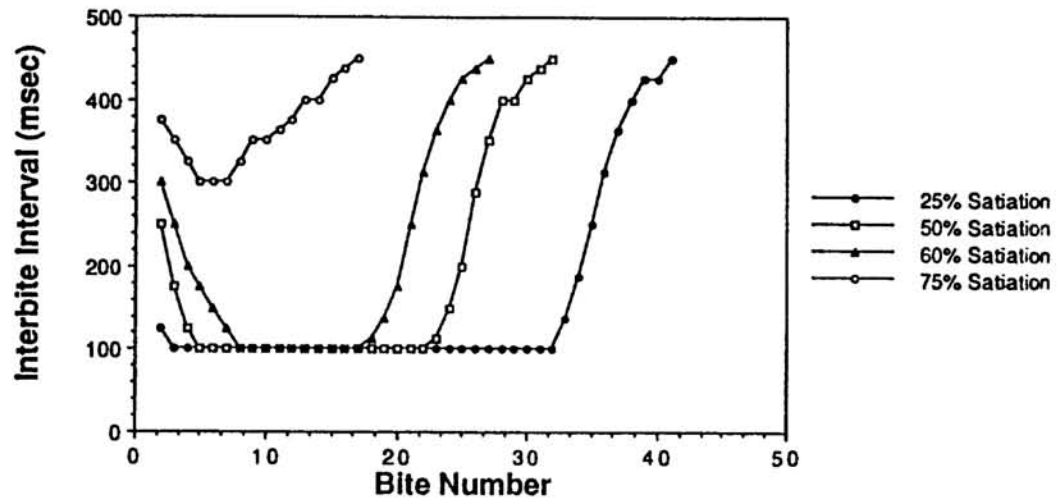

**Figure 3:** Build-Up of Arousal and Satiation

conditions described above (Figure 4).

An insect with a full complement of energy begins at the lower right-hand corner of the diagram, with maximum activity in ES and no activity in FA. As the insect's energy begins to fall, it moves to the left on the ES axis until the inhibition from ES is insufficient to hold FA below threshold. At this point, activity in FA begins to increase. Since the positive feedback loop is not yet active because no biting has occurred, a linear decrease in energy results in a linear increase in FA activity. If no food is consumed, the insect continues to move along this line toward the upper left of the diagram until its energy is exhausted.

However, if biting is triggered by the presence of food at the mouth, the relationship between FA and ES changes drastically. As the insect begins consuming food, activity in FA initially increases as arousal builds up, and then later decreases as the insect satiates. Each "bump" corresponds to the arousing effects on FA of one bite via the positive feedback loop and to the small increase of energy from the food consumed in that bite. Trajectories are shown for energy levels of 25%, 50%, 60%, 65%, and 75% of capacity. The shape of these trajectories depend upon the activity level of FA and the gain of the positive feedback loop in which it is embedded, both of which in turn depend upon the negative feedback from the energy sensor. We must therefore conclude that, even in this simple artificial insect, there is no single neural correlate to "hunger". Instead, this motivational state is the result of the complex dynamics of interaction between the feeding arousal neuron and the internal energy sensor.

## References

Beer, R. D. (1989). *Intelligence as Adaptive Behavior: An Experiment in Computational Neuroethology.* Ph.D. Dissertation, Dept. of Computer Engineering and Science, Case Western Reserve University. Also available as Technical Report TR

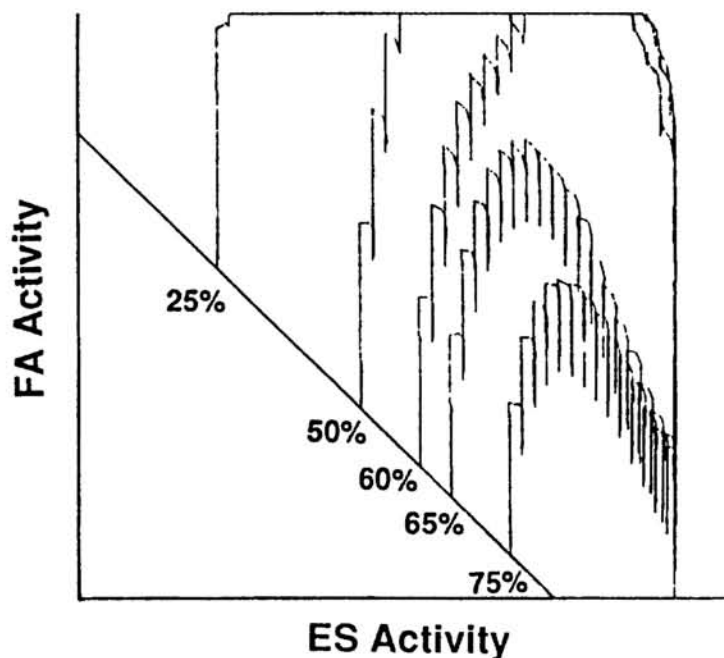

**Figure 4:** Phase Plot of FA vs. ES Activity

89-118, Center for Automation and Intelligent Systems Research.

Beer, R. D., Chiel, H. J. and Sterling, L. S. (1989). Heterogeneous Neural Networks for Adaptive Behavior in Dynamic Environments. In D.S. Touretzky (Ed.), *Advances in Neural Information Processing Systems 1* (pp. 577-585). San Mateo, CA: Morgan Kaufmann Publishers.

Chiel, H. J. and Beer, R. D. (1989). A lesion study of a heterogeneous neural network for hexapod locomotion. *Proceedings of the International Joint Conference on Neural Networks* (IJCNN 89), pp. 407-414.

Kupfermann, I. J. (1974). Feeding behavior in *Aplysia*: A simple system for the study of motivation. *Behavioral Biology* 10:1-26.

Susswein, A. J., Weiss, K. R. and Kupfermann, I. (1978). The effects of food arousal on the latency of biting in *Aplysia*. *J. Comp. Physiol.* 123:31-41.

Weiss, K. R., Chiel, II. J., Koch, U. and Kupfermann, I. (1986). Activity of an identified histaminergic neuron, and its possible role in arousal of feeding behavior in semi-intact *Aplysia*. *J. Neuroscience* 6(8):2403-2415.

Weiss, K. R., Chiel, II. J. and Kupfermann, I. (1986). Sensory function and gating of histaminergic neuron C2 in *Aplysia*. *J. Neuroscience* 6(8):2416-2426.